# *EyeGraph*: Modularity-aware Spatio Temporal Graph Clustering for Continuous Event-based Eye Tracking

**Nuwan Bandara[†], Thivya Kandappu[†], Argha Sen[*], Ila Gokarn[‡], Archan Misra[†]**

[†]Singapore Management University, [*]Indian Institute of Technology, Kharagpur
[‡]Singapore-MIT Alliance for Research and Technology (SMART)
{pmnsbandara, thivyak}@smu.edu.sg, arghasen10@gmail.com,
ila.gokarn@smart.mit.edu, archanm@smu.edu.sg

## Abstract

Continuous tracking of eye movement dynamics plays a significant role in developing a broad spectrum of human-centered applications, such as cognitive skills modeling, biometric user authentication, and foveated rendering. Recently neuromorphic cameras have garnered significant interest in the eye-tracking research community, owing to their sub-microsecond latency in capturing intensity changes resulting from eye movements. Nevertheless, the existing approaches for event-based eye tracking suffer from several limitations: dependence on RGB frames, label sparsity, and training on datasets collected in controlled lab environments that do not adequately reflect real-world scenarios. To address these limitations, in this paper, we propose a dynamic graph-based approach that uses the event stream for high-fidelity tracking of pupillary movement. We first present *EyeGraph*, a large-scale, multi-modal near-eye tracking dataset collected using a wearable event camera attached to a head-mounted device from 40 participants – the dataset was curated while mimicking in-the-wild settings, with variations in user movement and ambient lighting conditions. Subsequently, to address the issue of label sparsity, we propose an unsupervised topology-aware spatio-temporal graph clustering approach as a benchmark. We show that our unsupervised approach achieves performance comparable to more onerous supervised approaches while consistently outperforming the conventional clustering-based unsupervised approaches.

## 1 Introduction

Fine-grained, high-frequency eye tracking is increasingly of interest as an enabler of a wide variety of applications, such as biometric user authentication [30, 40, 57], foveated rendering for augmented and virtual reality [27, 43], and monitoring of cognitive attention/overload [15]. However, rapid and intricate eye movements [32] (with pupillary acceleration reaching values as high as $24,000°/s^2$ [1]), such as fixations (moments when the eyes are stationary and focused on a particular point), saccades (quick movements of both eyes between fixation points in the same direction), and microsaccades (small involuntary eye movements within fixation points) are difficult to capture with conventional RGB cameras [2, 35] due to their poor temporal resolution, susceptibility to motion blur, and constrained capability to accurately detect and track pupils under low lighting conditions. In this paper, we explore the possibility of using Neuromorphic event cameras [37, 20] as an alternative to traditional RGB-based eye tracking. Neuromorphic vision sensors capture changes in the visual scene asynchronously, only recording data when a significant event occurs, leading to a more efficient, high-frequency, and finer-grained depiction of eye movement dynamics.

Recent approaches for event-based eye-tracking predominantly adopt a supervised learning approach that uses either (i) RGB frames to guide pupil localization in event streams [59, 2, 73, 9] or (ii)

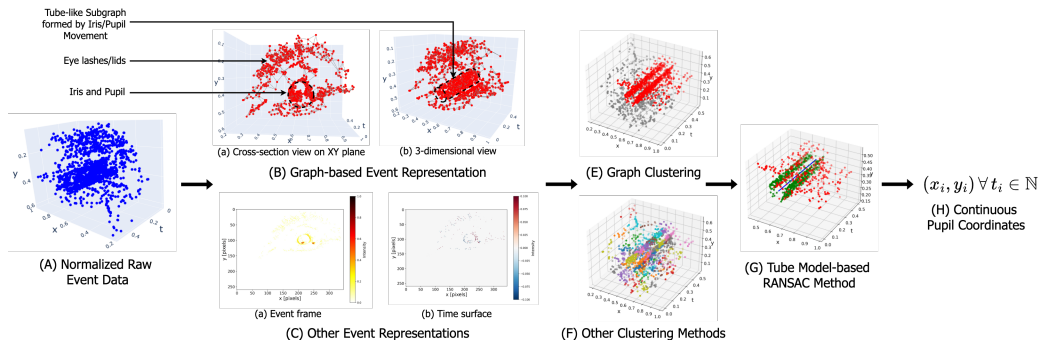

Figure 1: Broad overview of our approach, starting from (A) and follows the proposed steps: (B) in Section 5, (E) in Section 6 and (G) in Section 6 towards (H) continuous event-based eye tracking.

exclusive event-based pupil segmentation [35, 8]. Our main goal is to accurately and continuously record eye movements with high temporal resolution by (a) processing a stream of sparse events as asynchronously and temporally evolving graphs, and (b) adopting unsupervised modularity-aware graph representation learning to effectively cluster distinct ocular regions as a means for accurate and continuous pupil tracking. Our core idea is depicted in Figure 1.

In this paper, we specifically tackle four core shortcomings of event-based eye-tracking: (a) lack of event-based datasets that are representative of real-world scenarios (such as varying illuminance and user mobility – existing datasets are curated in a controlled lab environment with no head movements), (b) label sparsity (eye/gaze coordinates are only available on a fixed time interval, typically at coarser temporal resolution, e.g., EV-Eye [73] provides Points of Gaze at 100Hz), (c) dense 2D framed representation (events accumulated over a fixed time or event volume) that inadequately captures the underlying geometric, spatial, and temporal relationships, and (d) RGB-guided gaze inference (RGB cameras capture frames at fixed intervals, leading to a mismatch with the asynchronous nature of event-based sensors).

We make the following key contributions to address these limitations:

- To address the challenge (a), we present *EyeGraph*, a large-scale multi-modal near-eye tracking monocular dataset collected using a wearable event camera attached to a head-mounted device. It is important to note that the dataset was collected while mimicking in-the-wild settings, under varying ambient illuminance, and with individuals exhibiting unrestricted head and body movements.
- To tackle the challenges (b), (c), and (d) we adopt an unsupervised topology-aware graph-based approach. We propose (i) a novel temporally evolving dynamic graph representation for event-only eye tracking, and (ii) a novel topologically guided modularity-aware graph clustering approach that balances spatial proximity and temporal continuity of events. This approach ensures that the resulting clusters are accurately reflecting the sequential nature of eye movements.

To the best of our knowledge, our proposed benchmark is the first unsupervised event-based eye tracking in the literature.

## 2 Related Work

**Event-based Supervised Eye/Gaze Tracking:** Event cameras [37] mimic the human retina to record per-pixel changes in light intensity, yielding high $O(\mu s)$ temporal resolution, low $O(mW)$ power consumption, and asynchronous and sparse event streams. Recent works on eye/gaze tracking explore (i) hybrid RGB+Event approaches to pupil detection (with RGB frames) and tracking (with events) for near-eye gaze estimation [17, 73, 33, 71, 2], and (ii) pure-event approaches that aggregate events into framed representations for inference by DNN-based pupil segmentation and tracking [59, 34, 9, 8, 35, 64] and traditional computer vision methods like ellipse fitting for pupil-related events [35]. Hybrid approaches still rely on RGB frames, present lower perceptual throughput, and do not leverage the high temporal resolution of event data effectively. On the other hand, existing event-based processing suffers from challenges such as label sparsity, and inefficient framed representations.

**Graph-based Object Tracking:** Recent works have explored graph-based processing to maintain the sparse and asynchronous nature of event streams during processing and inference to improve *object tracking fidelity*. Early works in this paradigm construct *static, compact* graphs from event streams over distinct chunks of time, sacrificing the low-latency nature of event data in favour of aggregating information from events within each group or time period [5, 44] for more efficient graph processing. In contrast, recent works explore dynamic graph construction strategies such as (i) sliding window-based event-by-event graph construction [36], (ii) spatio-temporal graphs constructed from density-insensitive downsampled key-events [75], and (iii) "evolving" spatio-temporal graphs which restrict recomputation of network activations only to those nodes which are affected by incoming events [53]. These approaches reduce processing latency and computational complexity while maintaining tracking accuracy. However, Mondal et. al. [45] show how unsupervised clustering of events in graphs can support the detection of *distinct* moving objects, resulting in superior object tracking fidelity. To achieve efficient graph clustering, recent works explore strategies such as (i) unsupervised graph pooling [61, 7, 6, 69], (ii) autoencoders [41, 48, 29], and (iii) learnt patched representations [63, 72].

Our work cuts across the two fields of work to propose a novel dynamic graph construction mechanism with unsupervised *topological* clustering to efficiently isolate and track eye (pupillary) movement.

## 3 Motivation

### 3.1 Monocular Eye Tracking

While correlated, *pupil trajectory*, which is the movement of the pupil within the eye over time, and *gaze direction*, which is the direction in which a person looks relative to their environment, represent different aspects of eye movement. As an example, changes in lighting conditions or cognitive load can cause fluctuations in pupil movement without necessarily corresponding to changes in gaze direction, whereas reflexive eye movements, such as saccades or smooth pursuit, can cause rapid changes in gaze direction while the pupil trajectory remains relatively stable. Since our objective in this work is to track the reflexive, physiologically-driven spatiotemporal dynamics of the pupil, we focus on tracking the pupil's spatial coordinates in monocular fashion, rather than the trajectory of the gaze direction.

Our approach of tracking a single pupil is based on the assumption that users demonstrate ideal conjugate eye movements, reflecting synchronized ocular motions consistent with typical oculomotor function. Further, the broader understanding of saccadic and smooth pursuit eye movements suggests that both eyes move in a highly coordinated manner due to their control by shared neural circuits. To this end, many works in ophthalmology research discuss the stability and coordination of saccadic eye movements and imply that both eyes generally maintain similar velocity profiles, with the role of dominance not significantly altering this aspect of eye movement [46, 56, 32]. In addition, several prior works have used monocular eye tracking for: (a) 3D gaze tracking [74, 42], (b) emotion recognition [70, 67], (c) cognitive modelling [38, 25, 23], (d) virtual and mixed reality [55], and (e) user authentication [13].

### 3.2 Dynamic Graph Construction

Contrary to the conventional cameras (where the intensity of light across the visible spectrum incident on the sensor is captured at discrete points in time), event cameras or Dynamic Vision Sensors (DVS) only record changes in brightness (events) at each pixel asynchronously and with high temporal resolution, resulting in sparse data streams that encode motion and brightness changes in real-time. The event camera outputs a series of events on a per-pixel level – an event $e_i$ ($i \in \mathbb{N}$) is denoted by a tuple $(x_i, y_i, p_i, t_i)$, where $(x_i, y_i)$ denotes the corresponding pixel coordinates where the event is generated, $p_i$ represents the change in polarity (positive vs. negative), and $t_i$ is the time of the corresponding event.

Recent works on event processing involve Graph Neural Networks (GNNs) to process events as "static" spatio-temporal graphs [53, 5, 45, 36, 75]. These graphs are inherently "sparse", capturing the essential spatial and temporal relationships with a focus on efficient computation and representation. Inspired by the Hebbian learning principle, "pixels that fire together wire together", in this work, to efficiently process the event stream, we propose a dynamic and temporally *evolving* spatio-

temporal graph-based approach with adaptive edge construction. More specifically, the adaptive edge construction framework focuses on a Gaussian Mixture Model (GMM)-based soft clustering approach to spatially group distinct macroscopic parts of the human eye. Subsequently, the edges in the temporal plane are formed by connecting the nodes that are both spatially and temporally together. Our premise is that by preserving the local and global structure of the eye anatomy and its movements, the dynamic graphs can accurately represent the movement of various parts of the eye.

### 3.3 Unsupervised Topological Graph Clustering

Topological graph clustering partitions eye-tracking data into spatially and temporally coherent clusters, taking into account the modular structure of the underlying graph representation. It identifies densely connected subgraphs (clusters) that exhibit high within-cluster connectivity and low between-cluster connectivity. By considering both spatial proximity and temporal adjacency of nodes within the graph, the nodes within the same cluster are not only close in space but also temporally contiguous, reflecting the natural progression of gaze behavior over time. Further, we operate under the premise that each connected subgraph can represent a distinct anatomical or functional region of the eye, such as the pupil, upper eyelid, or a segment of the eyebrow. To be more specific, different parts of the eye exhibit (a) unique shapes, and (b) movement profiles. This allows the use of topological clustering on dynamically evolving graphs to (a) ensure that nodes within clusters are spatially close, reflecting physical proximity on the eye (i.e., shape of the eye anatomy), and (b) ensure clusters represent continuous movements by considering temporal order of events (i.e., distinct movement profiles of each eye region). To have a better visual understanding, in Figure 1(B), we show (i) the cross-section view on the spatial plane where the GMM-based soft clustering approach helps to identify the volume of events that are triggered by the movements of iris (i.e., the events forming a circle/oval contour), and (ii) the 3-dimensional view where we can witness that the temporal trajectory of iris movement is represented as a connected and directed sub-graph.

## 4 *EyeGraph*: A Large-Scale Mobile Event Dataset

### 4.1 Experimental Setup

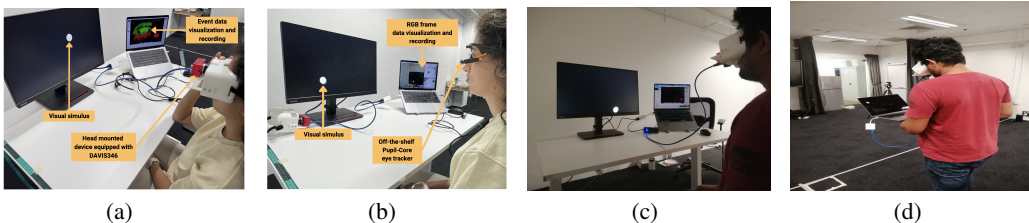

Figure 2: In-the-wild experiments under (a) lab settings; DAVIS$346$ (b) lab settings; Pupil-Core eye tracker, (c) varying illuminance, and (d) user mobility and head/body movements.

During the data collection process, the participants wear a custom-built head-mounted device (HMD) equipped with a DAVIS$346$ camera [24]. The HMD was secured around the forehead using a Velcro fastener. The camera is positioned adjacent to the right eye, while the participants are directed to track the visual stimuli using their left eye. To elicit natural eye movements, the visual stimulus appears at the top left corner of the screen and then moves continuously in random directions. To guide the gaze movement of the participants, we displayed the visual stimulus on a $1920 \times 1080$, 23.8-inch monitor. The distance between the monitor and the participant varied between $45cm$ and $50cm$, resulting in a field of view between $56° \times 34°$ and $62° \times 37°$, except when the participant moves freely. To collect reference for cross-modal investigations, the participants wear the off-the-shelf Pupil-Core eye tracker [26] at which their gaze is guided by replaying an identical visual stimulus.

To record eye-tracking data for a wider range of practical and in-the-wild conditions, as illustrated in Figure 2, we use three experimental setups: (i) *conventional lab settings* – the participant is seated in an office environment (default illuminance) while watching the visual stimulus on a screen. The participants can move their heads without maintaining a fixed/rigid posture, (ii) *changing ambient*

*illuminance*: the experiment is repeated both under regular lighting (348 Lux) and under lower illuminance (24 Lux), with the corresponding *near-eye* Lux values being 65 and 8 Lux, respectively, and (iii) *user mobility* – the participants are asked to move around freely within the lab while carrying a laptop ($3024 \times 1964$, 14-inch screen) that displays the visual stimuli, resulting in natural head and body movement.

## 4.2 Dataset Collection

Our participant pool consists of 40 participants, including 28 males and 12 females, representing diverse ethnic backgrounds (ages ranging from 21 to 32 years, $\mu = 26.08$ years and $\sigma = 2.99$) [1]. Prior to the data collection, the wearable devices were calibrated mechanically to ensure an optimal capture of each participant's eye region. The visual stimulus is a solid white circle (diameter 80 pixels) on a black background. While focusing their gaze on the white circle, each participant predominantly exhibited smooth pursuit and fixation states when the circle was moving smoothly. Similarly, saccadic states were triggered by the occasional discontinuous "jump" in the location of the white circle. The same experiment is repeated with variations in (i) the ambient illuminance from 348 Lux (default) to 24 Lux, and (ii) user movements.

## 4.3 Dataset Characteristics

To our knowledge, *EyeGraph* is the only dataset that comprehensively captures eye-tracking data under naturalistic indoor conditions. In Table 1, we present a comparative summary of *EyeGraph* vs. four publicly available event-based eye-tracking datasets, highlighting the distinctive attributes and advantages of each. Further, a detailed analysis on *EyeGraph* is presented in supplementary materials.

Table 1: Comparison of publicly available near-eye event datasets with *EyeGraph*

| Feature | EBV-Eye [2] | EV-Eye [73] | 3ET [9] | 3ET+ [64] | *EyeGraph* |
|---|---|---|---|---|---|
| Tracking End Goal | Gaze | Gaze | Pupil | Pupil | Pupil |
| Representation | 2D frame | 2D frame | 2D frame | 2D frame | Graph |
| Learning | supervised | supervised | supervised | supervised | unsupervised |
| Has Grayscale/RGB Frame Data? | ✓ | ✓ | ✗ | ✗ | ✓ |
| Is data from human participants? | ✓ | ✓ | ✗ | ✓ | ✓ |
| Is Monocular? | ✗ | ✗ | N/A | ✓ | ✓ |
| Is Multi-modal? | ✓ | ✓ | ✗ | ✗ | ✓ |
| Number of participants | 24 | 48 | N/A | 13 | 40 |
| Is head-movement allowed? | ✗ | ✗ | N/A | ✗ | ✓ |
| Accounts lighting changes? | ✗ | ✗ | ✗ | ✗ | ✓ |
| Accounts participant mobility? | ✗ | ✗ | N/A | ✗ | ✓ |

## 5 Dynamic Graph Construction

**Prerequisites**  A graph ($G$) is defined as $G = (V, E)$ with vertices $V = (v_1, ..., v_n)$; $|V| = n$ and edges $E = (e_1, ..., e_m) \subseteq V \times V$; $|E| = m$. We denote the $n \times n$ adjacency matrix (without self-loops) of $G$ by $\mathbf{A}$ where $A_{ij} = 1$ iff $\{v_i, v_j\} \in E$ given $i \neq j$, and $A_{ij} = 0$ otherwise. Further, each node is embedded with a $d-$dimensional feature vector $\mathbf{x}_{v_i} \in \mathbb{R}^d$ whereas each edge is embedded with a scalar feature $\mathbf{x}_{e_i} \in \mathbb{R}$. Further, the degree of a node $v_i$ is defined as its number of connections: $\sum_{j=1}^{n} A_{ij}$ and a graph is said to be directed iff $\forall v_i, v_j \in V; i, j \in \{k \in \mathbb{N} : k \leq n\}; i \neq j : \{v_i, v_j\} \in E$ where $\{v_i, v_j\}$ is an ordered pair.

**Problem Formulation**  Considering each event is encoded in a tuple: $(x_i, y_i, t_i, p_i) \, \forall \, i \in \mathbb{N}$, we represent raw events as a sparse and asynchronous spatio-temporal point cloud (as depicted in Figure 1) where events are represented as nodes in the graph. Therefore, the position of each node is denoted as $\mathbf{p}_{v_i} = [\lambda_1 t_i, \lambda_2 x_i, \lambda_3 y_i] \in \mathbb{R}^3$ and the corresponding node feature vector is represented as $[\lambda_1 t_i, \lambda_2 x_i, \lambda_3 y_i, p_i] \in \mathbb{R}^4$ where $\lambda_1, \lambda_2$ and $\lambda_3$ are normalization factors. This representation

allows us to effectively capture the temporal resolution of events (as opposed to the other event-based fixed representations depicted in Figure 1) resulting in efficient processing of the incoming events through sparse, but complete, graph updates [53]. Inspired by the Hebbian learning principle, "pixels that fire together wire together", our goal is to construct event-based dynamic graphs where the edges are formed to preserve the local and global structure of the eye's anatomical parts. This facilitates the tracking of each anatomical part's movement over time, represented as evolving snapshots in the temporal domain.

**Dynamic Theresholding-based Edge Construction Strategy**    The existing works for event-based vision tasks using GNNs [53, 21, 45, 36] mostly overlook the importance of edge construction mechanism, which is critical for preserving the object structure information [16, 60]. The conventional methods such as fixed-radius or k-nearest neighbours (kNN) are not specifically tailored to accommodate the unique characteristics of event vision (See Figures 3(a), 3(b), 3(c)). Hence, we construct edges based on a *dynamic threshold-based radius*, where the threshold changes based on the event volume and the movement dynamics.

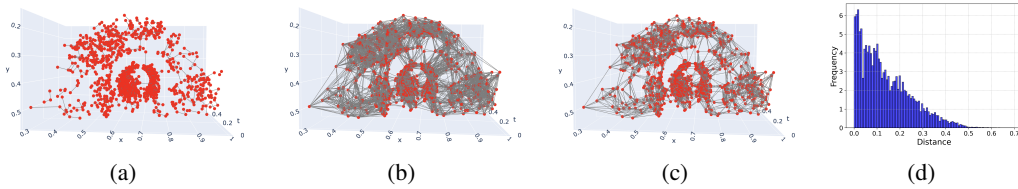

(a)                    (b)                    (c)                    (d)

Figure 3: An artificially set-threshold in radius graphs results in either (a) more disconnected components (i.e., global structure loss) or (b) more unintuitive edges (i.e., local structure loss) especially when the eye movement dynamics rapidly change. (c) kNN graphs are also susceptible to blur structure information of the events, due to the necessity to create edges up to an artificially-set threshold. (d) spatio-temporal distance distribution of an event volume.

To better understand the correlation between the events, in Figure 3(d), we depict the distribution of the Euclidean distances between spatio-temporal points. These distances can be represented as a mixture of Gaussian distributions, arising from the distinct eye parts and their varying movement profiles. We thus use Equation 1 for constructing the edges over a fixed event volume.

$$F(c, \mathbf{P}) = f^2_{k \in \mathbb{N} \text{ s.t. } 1 \leq k \leq c}(k, f^1_{\forall v_i, v_j \in V}[\|\mathbf{p}_{v_i} - \mathbf{p}_{v_j}\|]) \tag{1}$$

where $f^1(.)$ is the transformation function to retrieve the upper triangular matrix such that $D = [d_{ij}]$ iff $d_{ij} = 0 \, \forall i \geq j$ given $d_{ij} = \|\mathbf{p}_{v_i} - \mathbf{p}_{v_j}\|$ and $f^2(.)$ is the Gaussian mixture model (GMM) fitting function [66]: $\mathcal{N}(\mu_a, \sigma_a) = \sum_{a=1}^{k} \pi_a \mathcal{N}(x|\mu_a, \sigma_a)$, $1 \leq k \leq c$; with the objective of Bayesian information criterion ($BIC$) approaching minimum: $BIC[F(.)] \leq \delta$ (see supplementary materials for details). In addition, $c$ is the maximum number of clusters to be considered in GMM fitting which is heuristically determined as 5 due to the prominent anatomical clusters available in near-eye tracking: pupil, iris, lower and upper eyelids/lashes, and eyebrows. $\mathbf{P}$ denotes the set of node positions: $\mathbf{P} = (\mathbf{p}_{v_1}, ..., \mathbf{p}_{v_n})$.

After GMM fitting, the dynamic threshold for the accumulated event volume is set by considering the statistical relevance: $\xi_1 = \lambda \times min(\mu_i - 3\sigma_i)$ where $\mu_i$ and $\sigma_i$ are the mean and standard deviation of each fitted Gaussian distributions where the number of Gaussian distributions lies in $[1, c]$. Here, $\lambda$ is the scaling factor. Subsequently, this dynamic threshold is utilized in constructing radius-graph edges in the spatial plane: $\exists \{v_i, v_j\} \in E \, \forall t_i = t_j \pm \delta : \|\mathbf{p}_{v_i} - \mathbf{p}_{v_j}\| \leq \xi_1$ with a node degree condition: $\sum_{j=1}^{n} \mathbf{A}_{ij} \leq N$ to regularize the graph where $N$ is the allowed maximum degree. However, the edges in the temporal domain are constructed by following Equation 2 in a directed fashion to reflect the evolving nature of the graph in time.

$$\exists \{v_i, v_j\} \in E \, \forall t_i < t_j \text{ iff } \lambda_1(t_j - t_i) \leq \xi_2 \cap (x_j, y_j) \in \{(x_k, y_k)|k \in [i - \alpha, i + \alpha], \alpha \in \mathbb{N}\} \tag{2}$$

where $\xi_2$ and $\alpha$ are heuristically determined such that the evolving present and past events are well-connected, both spatially and temporally, in the node neighbourhood during the graph construction.

**Edge Feature Assignment**   In our approach, edge features are utilized to represent the relative movement profiles of eye parts. To capture the influence of historical events holistically, we adopt a Hawkes process-based [22, 31] attribution mechanism. The pseudo-code for the proposed edge feature assignment is depicted in Figure 4(a).

```python
def compute_edge_features(self, data, edge_index, num_divisions):
    # temporally segment the graph & count number of nodes in each segment
    node_counts, slot_width, min_value = self.divide_data(data, num_divisions)
    # normalize and update node counts with Hawkes effect of past events
    for i in range(len(node_counts)):
        current_count = node_counts[i]; updated_count = 1
        for j in range(1, min(i + 1, upper_bound)): # upper-bound for history
            updated_count += (node_counts[i-j]/(current_count+eps))*(decay_factor**j)
        updated_node_counts.append(updated_count)
    # assign updated node count as edge feature to every node in each segment
    for src, dst in edge_index.t().tolist():
        edge_features.append(updated_node_counts[min(int((data.pos[dst, 0] - \
                                 min_value)/slot_width), num_divisions-1)])
    return edge_features
```

```python
def ransac_tube_model(self, clusters):
    # select cluster(s) using PCA and ransac
    for cluster in clusters:
        tf_points = self.PCA(cluster, n_components = 2)
        ransac_reg.fit(tf_points); c_coords, r = self.PCA_inverse(ransac_reg)
        _, fit_error = self.compute_err(ransac_reg.in_mask, tf_points, c_coords, r)
        selected_clusters.append(cluster if fit_error < threshold_a)
    # derive best-cluster and its tube parameters using ransac-like pipeline
    for selected_cluster in selected_clusters:
        candidate_model = TubeModel(radius in radius_range)
        for _ in range(n_iterations):
            candidate_model.fit(points[random_sample])
            n_inliers = sum(candidate_model.predict(points)< threshold_b)
        best_model, best_inliers = candidate_model if n_inliers > best_inliers
    return best_cluster, centerline, radius if best_inliers in selected_cluster
```

(a)                                                                            (b)

Figure 4: PyTorch-style pseudo-codes for (a) Hawkes-based edge feature attribution, (b) RANSAC-based pupil coordinates estimation.

# 6   Unsupervised Topological Clustering

**Problem Formulation**   As empirically observed in Figure 1, event-based eye movement data suggests an underlying anatomical structure of the eye within the spatial plane. Simultaneously, it maintains the continuous movement dynamics of each component of the eye across the temporal domain. Consequently, it appears intuitive to use spatio-temporal clustering of the nodes within the graph representation, with the intent of extracting distinct movement profiles and dynamics for each anatomical part. To this end, our goal of clustering is to develop a graph partitioning function (without any label-support, i.e., unsupervised), $\Upsilon : V \mapsto \{1, ..., c\}$ s.t. $V_i = \{v_j : \Upsilon(v_j) = i\}$ to split the set of nodes into $c$ partitions via a graph encoder $f_\theta$ that maps the graph space ($G$) and the corresponding low-dimensional latent vector space: $f_\theta(G) = \mathbf{Z} \in \mathbb{R}^{n \times d_h}$ (where $\mathbf{Z} = \{z_i \mid i \in [1, n]\}$ and $d_h$ is the embedding dimension), ensuring the (i) anatomical topology of the eye with the temporal evolution are well-preserved while (ii) the cluster quality in terms of modularity measure is maximized.

**Variational Graph Autoencoder**   To learn the $f_\theta$, we implement a variational graph autoencoder (VGAE) [29] due to their demonstrated performance in highlighting the topology of the graphs through edge reconstruction. The encoder of the proposed VGAE is comprised of graph convolutional network layers [28] to generate node embeddings through the message passing rule: $\mathbf{X}^{(l+1)} = \eta(\widetilde{\mathbf{A}}\mathbf{X}^{(l)}\mathbf{W}^{(l)})$ for $l \in \{0, .., L-1\}$. In summary, the encoder can be modelled as $q(\mathbf{Z} \mid \mathbf{X}, \mathbf{A}) = \prod_{i=1}^{n} \mathcal{N}(z_i \mid \mu_i, diag(\sigma_i^2))$ while the decoder is $p(\mathbf{A} \mid \mathbf{Z}) = \prod_{i=1}^{n} \prod_{j=1}^{n} p(A_{ij} \mid z_i, z_j)$ with $p(A_{ij} = 1 \mid z_i, z_j) = \eta(z_i^\top z_j)$.

**Learning Objectives**   We propose a joint objective function to guide the model to learn topological information [29] embedded with the graph which maps to eye anatomy while maximizing the modularity [4] such that the edges fall within clusters are dense whereas edges between clusters are sparse. Therefore, we implement the weighted objective function in Equation 3.

$$L = \gamma_1 \mathbb{E}_{q(\mathbf{Z} \mid \mathbf{X}, \mathbf{A})}[\log p(\mathbf{A} \mid \mathbf{Z})] - \gamma_2 \frac{Tr(\mathbf{B}\mathbf{X}\mathbf{X}^\top)}{2m} - \gamma_3 KL[q(\mathbf{Z} \mid \mathbf{X}, \mathbf{A}) \| p(\mathbf{Z})] \qquad (3)$$

where $KL[q(.)\|p(.)]$, $\mathbf{B}$ and $\gamma_{i;i \in \{1,2,3\}}$ are the Kullback-Leibler (KL) divergence (between $q(.)$ and $p(.)$), modularity matrix and scaling factors respectively. Here, topological guidance and modularity maximization are achieved through the edge reconstruction loss (i.e., first term) and pairwise cluster membership loss (i.e., second term) respectively, while KL divergence (i.e., third term) works as a regularizer.

**Pupil Coordinates Estimation**   Through our empirical evaluations, we observe that the iris and pupil collectively exhibit a tunnel (i.e., a combination of cylinders and tori)-like movement profile in the spatio-temporal event cloud and thereby in the constructed graph representation (as depicted

in Figure 1). Therefore, after assigning the nodes in the graph into mutually-exclusive clusters and with the premise that the targeted movement profile of iris and pupil are separated into one cluster, we implement a custom tube-model-based random sample consensus (RANSAC) [18] method to estimate the center-line movement of the pupil. The pseudo-code of the proposed method is depicted in Figure 4(b).

# 7 Experiments and Results

## 7.1 Baseline Methods

We use the following baseline edge construction mechanisms to evaluate the performance on predicting the pupil coordinates: (i) **Fixed-radius graph** ($\exists \{v_i, v_j\} \in E$ iff $\|\mathbf{p}_{v_i} - \mathbf{p}_{v_j}\|_d \leq \xi$), (ii) **kNN graph**, (iii) **Nearest neighbour graph** (NN), (iv) **Furthest point sampling graph** (FPS) [51], and (v) **Gabriel graph** [10].

The following clustering (CL) and supervised (SP) baselines are compared against our proposed approach for graph clustering-based pupil coordinate estimation (see supplementary materials for details): (i) Using spatio-temporal $\mathbf{P}$ as features for CL: **k-means** (with local Lloyd algorithm [39] and k-means++ seeding strategy [3]), **Affinity Propagation** [19], **Meanshift** [11], **Spectral Clustering** [58, 12], **DBSCAN** [54]; (ii) Using graph structure for CL: **SBM** [50]; (iii) using spatio-temporal $\mathbf{P}$ as features and (graph) structure for CL: **Vanilla Graph Autoencoder** (GAE) [29], **GSCEvent-Mod** [45], **DMoN** [61], **DGI** [62]; (iv) Non-graph based pupil coordinate estimation methods using SP techniques: **MambaPupil** [65], **bigBrains** [49], **FreeEVs** [64], **GoSparse** [64], **Efficient** [64].

## 7.2 Other Datasets

In addition to the *EyeGraph* dataset, we evaluate the performance of dynamic graph construction and graph clustering techniques on the publicly available EBV-Eye dataset [2]. EBV-Eye does not contain pupil coordinate labels as the dataset end-goal is gaze tracking. To evaluate the accuracy of our proposed unsupervised pupil localization, we use 3ET+ dataset [64]. This dataset was recently used in the AIS 2024 challenge [64] and we use several supervised methods proposed as part of the challenge to compare the performance of *EyeGraph*. 3ET+ consists of human-labelled ground-truths for pupil coordinates at $100Hz$.

## 7.3 Evaluation Metrics

Since our goal is to build modularity and topology-aware clustering capability where each cluster represents a distinct anatomical region of the eye, we employ the clustering quality as the primary evaluation target. Therefore, we utilize four distinct metrics for evaluating the cluster quality: (i) mean Silhouette coefficient (SC) [52], (ii) Davies-Bouldin score (DB) [14], (iii) modularity (Mo) [47], and (iv) conductance (Cd) [68]. Further, to compare with supervised approaches for pupil coordinate estimation, we use the p-accuracy [64], mean Euclidean ($l_2$) and Manhattan ($l_1$) distances following the literature.

## 7.4 Results

As tabulated in Table 2, our method for dynamic graph construction achieves superior or comparable performance across all evaluation metrics and various datasets. We observe that our method achieves higher SC and Mo scores, and lower Cd scores, indicating that dynamic graph construction helps in accurately representing the event-based eye region data. Similarly, in Table 3, for graph clustering, our proposed unsupervised topology-guided clustering consistently achieves higher Mo scores, hinting that the resultant spatio-temporal clusters are well separated. Further, in Table 4, we compare *EyeGraph* approach against supervised and standard unsupervised methods in estimating the pupil coordinates. While *EyeGraph's* $p10$ accuracy is $\approx 8\%$ lower than the best performing supervised approach proposed in [64], it performs superior among the unsupervised methods. In addition, we can see that the incorporation of our clustering approach is beneficial even to the DMoN approach, as it helps achieve 14% improvement in the p10 value. Further, since our dataset is collected using a mobile setup, accounting varying mobility and ambient lighting conditions, we evaluate the robustness of the proposed method in compared with the existing supervised methods (in Table 4). We see that,

even under poor lighting and conditions that include severe motion artifacts, we achieve an accuracy of 93.67%, indicating its capability to accurately represent and track event-based eye movement data. In comparison, even though MambaPupil achieves slightly better performance i.e., 2% improvement in p10 value, it is noteworthy that our approach does not demand any extensive labeling effort to achieve comparable performance.

Table 2: Performance of dynamic graph construction on *EyeGraph* and EBV-Eye using CL metrics. $^\star$3d spatio-temporal edges; instead of the proposed dual-step edges.

| | *EyeGraph* | | | | EBV-Eye | | | |
|---|---|---|---|---|---|---|---|---|
| Method | SC↑ | DB↓ | Mo↑ | Cd↓ | SC↑ | DB↓ | Mo↑ | Cd↓ |
| Radius ($\xi = 10^{-4}$) | 0.55 | 0.99 | 61.34 | 14.89 | 0.47 | 0.93 | 63.43 | 18.05 |
| Radius ($\xi = 10^{-1}$) | 0.53 | 0.92 | 62.45 | 14.90 | 0.42 | 0.99 | 63.85 | 13.14 |
| Radius$^\star$ ($\xi_{3d} = 10^{-3}$) | 0.64 | 0.97 | 68.80 | 12.45 | 0.49 | **0.90** | 67.86 | 10.71 |
| kNN ($k = 8$) | 0.62 | **0.89** | 69.30 | 16.34 | 0.43 | 1.02 | 70.11 | 6.21 |
| NN | 0.61 | 1.02 | 68.30 | 15.74 | 0.52 | 0.91 | 63.26 | 6.82 |
| FPS [51] | 0.52 | 1.03 | 60.54 | 15.49 | 0.41 | 1.34 | 68.45 | 15.87 |
| Gabriel [10] | 0.59 | 1.01 | 63.23 | 12.10 | 0.52 | 1.04 | 72.45 | 20.69 |
| Ours | **0.66** | 0.91 | **69.34** | **11.30** | **0.54** | **0.90** | **75.70** | **5.07** |

Table 3: Performance of the proposed graph clustering on *EyeGraph* and EBV-Eye using CL metrics

| | *EyeGraph* | | | | EBV-Eye | | | |
|---|---|---|---|---|---|---|---|---|
| Method | SC↑ | DB↓ | Mo↑ | Cd↓ | SC↑ | DB↓ | Mo↑ | Cd↓ |
| kmeans [39, 3] | 0.30 | 0.99 | 49.45 | 30.23 | 0.31 | 1.56 | 54.56 | 20.45 |
| Affinity [19] | 0.31 | 1.20 | 40.67 | 32.37 | 0.29 | 1.67 | 50.34 | 27.88 |
| Meanshift [11] | 0.33 | 1.33 | 40.50 | 28.64 | 0.40 | 1.34 | 55.89 | 19.80 |
| Spectral [58, 12] | 0.39 | 1.44 | 52.34 | 27.46 | 0.36 | 1.48 | 42.67 | 17.88 |
| DBSCAN [54] | 0.40 | 1.18 | 55.34 | 20.76 | 0.41 | 1.62 | 60.24 | 15.78 |
| SBM [50] | 0.45 | 1.20 | 60.36 | 17.77 | **0.56** | 1.22 | 62.88 | 14.90 |
| GAE [29] | 0.53 | 1.03 | 57.90 | 18.44 | 0.51 | 1.20 | 65.65 | 16.68 |
| GSCEventMod [45] | 0.51 | 1.00 | 62.67 | 12.07 | 0.50 | 1.02 | 67.88 | 10.34 |
| DGI [62] | 0.67 | 0.93 | 67.04 | **10.55** | 0.50 | 1.30 | 70.34 | 5.46 |
| DMoN [61] | **0.68** | **0.90** | 68.80 | 13.00 | 0.54 | **0.82** | 69.46 | 9.36 |
| Ours | 0.66 | 0.91 | **69.34** | 11.30 | 0.54 | 0.90 | **75.70** | **5.07** |

## 8 Discussion and Conclusion

**Limitations** While our results show the superior performance of *EyeGraph* in unsupervised eye-tracking, we identify the following open areas to be investigated. *EyeGraph* currently uses wearable-based near-eye tracking of only one eye, and thus cannot directly take advantage of gaze-related features such as saccades and fixation. Our benchmark technique assumes that the users typically tend to exhibit conjugate eye movement, moving both eyes in tandem having similar velocity profiles. However, it is possible that microscopic distinctions may exist between the pupillary movements of the right and left eyes, perhaps because of the differences in ocular muscle strength (most people have one dominant eye). In addition, we recognize the critical importance of binocular eye tracking data to support physiological conditions/medical diagnostics use cases, such as capturing neurodevelopmental disorders. Conversely, monocular tracking proves to be sufficiently effective in applications where depth perception is not a priority, such as user authentication, human-computer interaction, and emotion recognition. Further, the users are to follow the visual stimuli that randomly move across the screen in various directions. However, to mimic more in-the-wild settings, the data stimuli should involve natural cues such as interacting in the physical world, reading articles etc.

**Future Works and Applications** We are working to augment our dataset, and/or release new iterations, with more naturalistic, but visual stimuli-driven, in-the-wild studies in both indoor and outdoor environments so as to capture finer-grained continuous variation in illuminance, and during

Table 4: Performance of proposed unsupervised pipeline for pupil coordinates estimation across 3ET+ and our dataset with respect to p-score and distance-based evaluation metrics. ⋆ See supplementary materials for details.

| Category | Method | 3ET+ | | | | |
| --- | --- | --- | --- | --- | --- | --- |
| | | p10↑ | p5↑ | p1↑ | $l_2 \downarrow$ | $l_1 \downarrow$ |
| Supervised | MambaPupil [65] | **99.42** | 97.05 | 33.75 | 1.67 | 2.11 |
| | FreeEVs [64] | 99.26 | 96.31 | 23.91 | 2.03 | 2.56 |
| | bigBrains [49] | 99.00 | **97.79** | **45.50** | **1.44** | **1.82** |
| | GoSparse [64] | 99.00 | 77.20 | 7.32 | 3.51 | 4.63 |
| | Efficient [64] | 97.95 | 80.67 | 7.79 | 3.51 | 4.43 |
| Unsupervised | kmeans-1⋆ | 64.90 | 54.46 | 7.05 | 10.45 | 12.33 |
| | kmeans-2⋆ | 81.45 | 74.56 | 8.45 | 6.78 | 7.98 |
| | DMoN [61]⋆ | 77.45 | 75.07 | 10.20 | 8.36 | 9.66 |
| | Ours | **91.45** | **89.22** | **28.34** | **3.88** | **4.24** |
| *With varying ambient lighting and mobility conditions* | | | | | | |
| | | *EyeGraph* ⋆ | | | | |
| Supervised | MambaPupil [65] | 95.88 | 87.04 | 40.77 | 3.11 | 3.33 |
| Unsupervised | kmeans-1⋆ | 50.45 | 43.56 | 5.89 | 7.46 | 14.45 |
| | kmeans-2⋆ | 86.32 | 79.44 | 20.32 | 4.90 | 6.46 |
| | Ours | **93.67** | **91.78** | **31.90** | **3.54** | **3.98** |

diverse set of physical activities. In addition, in our future data collection efforts, we plan to include more natural cue-based stimuli, via real-world interaction scenarios instead of the utilized controlled visual stimuli. Further, we believe that *EyeGraph* dataset will also be instrumental in studying fine-grained pupillary movements for diverse applications such as continuous biometric authentication and affective-cognitive modelling. In contrast, we hope that our *EyeGraph* methods will also be useful in other domains such as automotive vision and robotics.

**Conclusion**   In this paper, we present *EyeGraph*, a large-scale multi-modal near-eye tracking dataset collected using a wearable event camera attached to a head-mounted device. We then propose an unsupervised topology-aware spatio-temporal graph clustering approach as a benchmark. Using extensive evaluations, we show that our unsupervised approach achieves comparable performance against the supervised approaches while consistently outperforming the conventional clustering approaches.

**Acknowledgements**   This work was supported by both the Ministry of Education (MOE) Academic Research Fund (AcRF) Tier 1 grant (Grant ID: 22-SIS-SMU-044), and by the National Research Foundation, Prime Minister's Office, Singapore under its NRF Investigatorship grant (NRF-NRFI05-2019-0007). Any opinions, findings and conclusions or recommendations expressed in this material are those of the author(s) and do not reflect the views of National Research Foundation, Singapore.

## Footnotes

[1]Our institution's IRB approved our data collection

## References

[1] R. A. Abrams, D. E. Meyer, and S. Kornblum. Speed and accuracy of saccadic eye movements: characteristics of impulse variability in the oculomotor system. *Journal of Experimental Psychology: Human Perception and Performance*, 15(3):529, 1989.

[2] A. N. Angelopoulos, J. N. Martel, A. P. Kohli, J. Conradt, and G. Wetzstein. Event-based near-eye gaze tracking beyond 10,000 Hz. *IEEE Transactions on Visualization and Computer Graphics*, 27(5):2577–2586, 2021.

[3] D. Arthur, S. Vassilvitskii, et al. k-means++: The advantages of careful seeding. In *Soda*, volume 7, pages 1027–1035, 2007.

[4] A. Bhowmick, M. Kosan, Z. Huang, A. Singh, and S. Medya. DGCLUSTER: A neural framework for attributed graph clustering via modularity maximization. In *Proceedings of the AAAI Conference on Artificial Intelligence*, volume 38, pages 11069–11077, 2024.

[5] Y. Bi, A. Chadha, A. Abbas, E. Bourtsoulatze, and Y. Andreopoulos. Graph-based object classification for neuromorphic vision sensing. In *Proceedings of the IEEE/CVF International Conference on Computer Vision*, pages 491–501, 2019.

[6] F. M. Bianchi, D. Grattarola, and C. Alippi. Mincut pooling in graph neural networks. 2019.

[7] F. M. Bianchi, D. Grattarola, and C. Alippi. Spectral clustering with graph neural networks for graph pooling, 2020.

[8] P. Bonazzi, S. Bian, G. Lippolis, Y. Li, S. Sheik, and M. Magno. Retina : Low-power eye tracking with event camera and spiking hardware, 2024.

[9] Q. Chen, Z. Wang, S.-C. Liu, and C. Gao. 3ET: Efficient event-based eye tracking using a change-based convlstm network. In *2023 IEEE Biomedical Circuits and Systems Conference (BioCAS)*, pages 1–5. IEEE, 2023.

[10] J. Choo, R. Jiamthapthaksin, C. S. Chen, O. U. Celepcikay, C. Giusti, and C. F. Eick. MOSAIC: A proximity graph approach for agglomerative clustering. In *Data Warehousing and Knowledge Discovery: 9th International Conference, DaWaK 2007, Regensburg Germany, September 3-7, 2007. Proceedings 9*, pages 231–240. Springer, 2007.

[11] D. Comaniciu and P. Meer. Mean shift: A robust approach toward feature space analysis. *IEEE Transactions on Pattern Analysis and Machine Intelligence*, 24(5):603–619, 2002.

[12] A. Damle, V. Minden, and L. Ying. Simple, direct and efficient multi-way spectral clustering. *Information and Inference: A Journal of the IMA*, 8(1):181–203, 2019.

[13] I. Das, R. Das, S. Singh, A. Banerjee, M. G. Mohiuddin, and A. Chowdhury. Design and implementation of eye pupil movement based pin authentication system. In *2020 IEEE VLSI DEVICE CIRCUIT AND SYSTEM (VLSI DCS)*, pages 1–6, 2020. doi: 10.1109/VLSIDCS47293.2020.9179933.

[14] D. L. Davies and D. W. Bouldin. A cluster separation measure. *IEEE Transactions on Pattern Analysis and Machine Intelligence*, (2):224–227, 1979.

[15] A. T. Duchowski, K. Krejtz, I. Krejtz, C. Biele, A. Niedzielska, P. Kiefer, M. Raubal, and I. Giannopoulos. The index of pupillary activity: Measuring cognitive load vis-à-vis task difficulty with pupil oscillation. In *Proceedings of the 2018 CHI conference on human factors in computing systems*, pages 1–13, 2018.

[16] S. Faisal, G. Tziantzioulis, A. M. Gok, N. Hardavellas, S. Ogrenci-Memik, and S. Parthasarathy. Edge importance identification for energy efficient graph processing. In *2015 IEEE International Conference on Big Data (Big Data)*, pages 347–354. IEEE, 2015.

[17] Y. Feng, N. Goulding-Hotta, A. Khan, H. Reyserhove, and Y. Zhu. Real-time gaze tracking with event-driven eye segmentation. In *2022 IEEE Conference on Virtual Reality and 3D User Interfaces (VR)*, pages 399–408. IEEE, 2022.

[18] M. A. Fischler and R. C. Bolles. Random sample consensus: a paradigm for model fitting with applications to image analysis and automated cartography. *Communications of the ACM*, 24(6):381–395, 1981.

[19] B. J. Frey and D. Dueck. Clustering by passing messages between data points. *Science*, 315 (5814):972–976, 2007.

[20] G. Gallego, T. Delbrück, G. Orchard, C. Bartolozzi, B. Taba, A. Censi, S. Leutenegger, A. J. Davison, J. Conradt, K. Daniilidis, et al. Event-based vision: A survey. *IEEE Transactions on Pattern Analysis and Machine Intelligence*, 44(1):154–180, 2020.

[21] D. Gehrig and D. Scaramuzza. Low-latency automotive vision with event cameras. *Nature*, 629 (8014):1034–1040, 2024.

[22] A. G. Hawkes. Spectra of some self-exciting and mutually exciting point processes. *Biometrika*, 58(1):83–90, 1971.

[23] T. Heinen and P. M. Vinken. Monocular and binocular vision in the performance of a complex skill. *Journal of Sports Science & Medicine*, 10(3):520, 2011.

[24] Inivation AG. *DAVIS346b*. Available at `https://inivation.com/wp-content/uploads/2019/08/DAVIS346.pdf`, Accessed: October 31, 2024.

[25] J. Johansson, T. Pansell, J. Ygge, and G. Ö. Seimyr. Monocular and binocular reading performance in subjects with normal binocular vision. *Clinical and Experimental Optometry*, 97(4): 341–348, 2014.

[26] M. Kassner, W. Patera, and A. Bulling. Pupil: an open source platform for pervasive eye tracking and mobile gaze-based interaction. In *Proceedings of the 2014 ACM international joint conference on pervasive and ubiquitous computing: Adjunct publication*, pages 1151–1160, 2014.

[27] J. Kim, Y. Jeong, M. Stengel, K. Aksit, R. A. Albert, B. Boudaoud, T. Greer, J. Kim, W. Lopes, Z. Majercik, et al. Foveated AR: dynamically-foveated augmented reality display. *ACM Trans. Graph.*, 38(4):99–1, 2019.

[28] T. N. Kipf and M. Welling. Semi-supervised classification with graph convolutional networks. *arXiv preprint arXiv:1609.02907*, 2016.

[29] T. N. Kipf and M. Welling. Variational graph auto-encoders. *arXiv preprint arXiv:1611.07308*, 2016.

[30] O. V. Komogortsev, S. Jayarathna, C. R. Aragon, and M. Mahmoud. Biometric identification via an oculomotor plant mathematical model. In *Proceedings of the 2010 Symposium on Eye-Tracking Research & Applications*, pages 57–60, 2010.

[31] P. J. Laub, Y. Lee, and T. Taimre. *The elements of Hawkes processes*. Springer, 2021.

[32] R. J. Leigh and D. S. Zee. *The neurology of eye movements*. Oxford University Press, USA, 2015.

[33] J. Li, Z. Zhu, J. Hou, J. Hou, and J. Wu. Denoising distillation makes event-frame transformers as accurate gaze trackers. *arXiv preprint arXiv:2404.00548*, 2024.

[34] N. Li, A. Bhat, and A. Raychowdhury. E-track: Eye tracking with event camera for extended reality (xr) applications. In *2023 IEEE 5th International Conference on Artificial Intelligence Circuits and Systems (AICAS)*, pages 1–5. IEEE, 2023.

[35] N. Li, M. Chang, and A. Raychowdhury. E-gaze: Gaze estimation with event camera. *IEEE Transactions on Pattern Analysis and Machine Intelligence*, 2024.

[36] Y. Li, H. Zhou, B. Yang, Y. Zhang, Z. Cui, H. Bao, and G. Zhang. Graph-based asynchronous event processing for rapid object recognition. In *Proceedings of the IEEE/CVF International Conference on Computer Vision*, pages 934–943, 2021.

[37] P. Lichtsteiner, C. Posch, and T. Delbruck. A $128 \times 128$ 120 db $15\mu s$ latency asynchronous temporal contrast vision sensor. *IEEE journal of solid-state circuits*, 43(2):566–576, 2008.

[38] Y. Lim, A. Gardi, N. Pongsakornsathien, R. Sabatini, N. Ezer, and T. Kistan. Experimental characterisation of eye-tracking sensors for adaptive human-machine systems. *Measurement*, 140:151–160, 2019.

[39] S. Lloyd. Least squares quantization in PCM. *IEEE transactions on information theory*, 28(2): 129–137, 1982.

[40] D. Lohr and O. V. Komogortsev. Eye know you too: Toward viable end-to-end eye movement biometrics for user authentication. *IEEE Transactions on Information Forensics and Security*, 17:3151–3164, 2022.

[41] Y. Ma, H. He, Z. Lei, and Z. Niu. Masked autoencoder for graph clustering without pre-defined cluster number k. *arXiv preprint arXiv:2401.04741*, 2024.

[42] M. Mansouryar, J. Steil, Y. Sugano, and A. Bulling. 3D gaze estimation from 2D pupil positions on monocular head-mounted eye trackers. In *Proceedings of the ninth biennial ACM symposium on eye tracking research & applications*, pages 197–200, 2016.

[43] S. L. Matthews, A. Uribe-Quevedo, and A. Theodorou. Rendering optimizations for virtual reality using eye-tracking. In *2020 22nd symposium on virtual and augmented reality (SVR)*, pages 398–405. IEEE, 2020.

[44] A. Mitrokhin, Z. Hua, C. Fermuller, and Y. Aloimonos. Learning visual motion segmentation using event surfaces. In *Proceedings of the IEEE/CVF Conference on Computer Vision and Pattern Recognition*, pages 14414–14423, 2020.

[45] A. Mondal, J. H. Giraldo, T. Bouwmans, A. S. Chowdhury, et al. Moving object detection for event-based vision using graph spectral clustering. In *Proceedings of the IEEE/CVF International Conference on Computer Vision*, pages 876–884, 2021.

[46] A. P. Morris, C. D. Chambers, and J. B. Mattingley. Parietal stimulation destabilizes spatial updating across saccadic eye movements. *Proceedings of the National Academy of Sciences*, 104(21):9069–9074, 2007.

[47] M. E. Newman. Modularity and community structure in networks. *Proceedings of the national academy of sciences*, 103(23):8577–8582, 2006.

[48] S. Pan, R. Hu, G. Long, J. Jiang, L. Yao, and C. Zhang. Adversarially regularized graph autoencoder for graph embedding, 2019.

[49] Y. R. Pei, S. Brüers, S. Crouzet, D. McLelland, and O. Coenen. A lightweight spatiotemporal network for online eye tracking with event camera. *arXiv preprint arXiv:2404.08858*, 2024.

[50] T. P. Peixoto. Efficient monte carlo and greedy heuristic for the inference of stochastic block models. *Physical Review E*, 89(1):012804, 2014.

[51] C. R. Qi, L. Yi, H. Su, and L. J. Guibas. Pointnet++: Deep hierarchical feature learning on point sets in a metric space. *Advances in Neural Information Processing Systems*, 30, 2017.

[52] P. J. Rousseeuw. Silhouettes: a graphical aid to the interpretation and validation of cluster analysis. *Journal of computational and applied mathematics*, 20:53–65, 1987.

[53] S. Schaefer, D. Gehrig, and D. Scaramuzza. AEGNN: Asynchronous event-based graph neural networks. In *Proceedings of the IEEE/CVF Conference on Computer Vision and Pattern Recognition*, pages 12371–12381, 2022.

[54] E. Schubert, J. Sander, M. Ester, H. P. Kriegel, and X. Xu. DBSCAN revisited, revisited: why and how you should (still) use DBSCAN. *ACM Transactions on Database Systems (TODS)*, 42 (3):1–21, 2017.

[55] I. Schuetz and K. Fiehler. Eye tracking in virtual reality: Vive pro eye spatial accuracy, precision, and calibration reliability. *Journal of Eye Movement Research*, 15(3), 2022.

[56] A. C. Schütz, D. I. Braun, and K. R. Gegenfurtner. Eye movements and perception: A selective review. *Journal of vision*, 11(5):9–9, 2011.

[57] A. Sen, N. Bandara, I. Gokarn, T. Kandappu, and A. Misra. EyeTrAES: fine-grained, low-latency eye tracking via adaptive event slicing. *arXiv preprint arXiv:2409.18813*, 2024.

[58] J. Shi and J. Malik. Normalized cuts and image segmentation. *IEEE Transactions on Pattern Analysis and Machine Intelligence*, 22(8):888–905, 2000.

[59] T. Stoffregen, H. Daraei, C. Robinson, and A. Fix. Event-based kilohertz eye tracking using coded differential lighting. In *Proceedings of the IEEE/CVF Winter Conference on Applications of Computer Vision*, pages 2515–2523, 2022.

[60] S. H. Tanneru. Edge importance scores for editing graph topology to preserve fairness. *ICML 2023 2nd Annual Topology, Algebra, and Geometry in Machine Learning Workshop*, 2023.

[61] A. Tsitsulin, J. Palowitch, B. Perozzi, and E. Müller. Graph clustering with graph neural networks. *Journal of Machine Learning Research*, 24(127):1–21, 2023.

[62] P. Velickovic, W. Fedus, W. L. Hamilton, P. Liò, Y. Bengio, and R. D. Hjelm. Deep graph infomax. *ICLR (Poster)*, 2(3):4, 2019.

[63] P. Veličković, W. Fedus, W. L. Hamilton, P. Liò, Y. Bengio, and R. D. Hjelm. Deep graph infomax. 2018.

[64] Z. Wang, C. Gao, Z. Wu, M. V. Conde, R. Timofte, S.-C. Liu, Q. Chen, Z.-j. Zha, W. Zhai, H. Han, et al. Event-based eye tracking. AIS 2024 challenge survey. *arXiv preprint arXiv:2404.11770*, 2024.

[65] Z. Wang, Z. Wan, H. Han, B. Liao, Y. Wu, W. Zhai, Y. Cao, and Z.-j. Zha. Mambapupil: Bidirectional selective recurrent model for event-based eye tracking. *arXiv preprint arXiv:2404.12083*, 2024.

[66] C. Williams and C. Rasmussen. Gaussian processes for regression. *Advances in Neural Information Processing Systems*, 8, 1995.

[67] H. Wu, J. Feng, X. Tian, E. Sun, Y. Liu, B. Dong, F. Xu, and S. Zhong. EMO: Real-time emotion recognition from single-eye images for resource-constrained eyewear devices. In *Proceedings of the 18th International Conference on Mobile Systems, Applications, and Services*, pages 448–461, 2020.

[68] J. Yang and J. Leskovec. Defining and evaluating network communities based on ground-truth. In *Proceedings of the ACM SIGKDD Workshop on Mining Data Semantics*, pages 1–8, 2012.

[69] Z. Ying, J. You, C. Morris, X. Ren, W. Hamilton, and J. Leskovec. Hierarchical graph representation learning with differentiable pooling. *Advances in Neural Information Processing Systems*, 31, 2018.

[70] H. Zhang, J. Zhang, B. Dong, P. Peers, W. Wu, X. Wei, F. Heide, and X. Yang. In the blink of an eye: Event-based emotion recognition. In *ACM SIGGRAPH 2023 Conference Proceedings*, pages 1–11, 2023.

[71] T. Zhang, Y. Shen, G. Zhao, L. Wang, X. Chen, L. Bai, and Y. Zhou. Swift-Eye: Towards anti-blink pupil tracking for precise and robust high-frequency near-eye movement analysis with event cameras. *IEEE Transactions on Visualization and Computer Graphics*, 2024.

[72] X. Zhang, H. Liu, Q. Li, and X.-M. Wu. Attributed graph clustering via adaptive graph convolution. *arXiv preprint arXiv:1906.01210*, 2019.

[73] G. Zhao, Y. Yang, J. Liu, N. Chen, Y. Shen, H. Wen, and G. Lan. EV-Eye: Rethinking high-frequency eye tracking through the lenses of event cameras. *Advances in Neural Information Processing Systems*, 36, 2024.

[74] W. Zhu and H. Deng. Monocular free-head 3d gaze tracking with deep learning and geometry constraints. In *Proceedings of the IEEE International Conference on Computer Vision*, pages 3143–3152, 2017.

[75] Z. Zhu, J. Hou, and X. Lyu. Learning graph-embedded key-event back-tracing for object tracking in event clouds. *Advances in Neural Information Processing Systems*, 35:7462–7476, 2022.

